# Maximum Margin Multi-Label Structured Prediction

**Christoph H. Lampert**
IST Austria (Institute of Science and Technology Austria)
Am Campus 1, 3400 Klosterneuburg, Austria
http://www.ist.ac.at/~chl    chl@ist.ac.at

## Abstract

We study *multi-label prediction* for *structured output sets*, a problem that occurs, for example, in object detection in images, secondary structure prediction in computational biology, and graph matching with symmetries. Conventional multi-label classification techniques are typically not applicable in this situation, because they require explicit enumeration of the label set, which is infeasible in case of structured outputs. Relying on techniques originally designed for single-label structured prediction, in particular structured support vector machines, results in reduced prediction accuracy, or leads to infeasible optimization problems.

In this work we derive a maximum-margin training formulation for multi-label structured prediction that remains computationally tractable while achieving high prediction accuracy. It also shares most beneficial properties with single-label maximum-margin approaches, in particular formulation as a convex optimization problem, efficient working set training, and PAC-Bayesian generalization bounds.

## 1 Introduction

The recent development of *conditional random fields (CRFs)* [1], *max-margin Markov networks (M3Ns)* [2], and *structured support vector machines (SSVMs)* [3] has triggered a wave of interest in the prediction of complex outputs. Typically, these are formulated as graph labeling or graph matching tasks in which each input has a unique correct output. However, not all problems encountered in real applications are reflected well by this assumption: *machine translation* in natural language processing, *secondary structure prediction* in computational biology, and *object detection* in computer vision are examples of tasks in which more than one prediction can be "correct" for each data sample, and that are therefore more naturally formulated as multi-label prediction tasks.

In this paper, we study *multi-label structured prediction*, defining the task and introducing the necessary notation in Section 2. Our main contribution is a formulation of a maximum-margin training problem, named MLSP, which we introduce in Section 3. Once trained it allows the prediction of multiple structured outputs from a single input, as well as abstaining from a decision. We study the generalization properties of MLSP in form of a generalization bound in Section 3.2, and we introduce a working set optimization procedure in Section 3.3. The main insights from these is that MLSP behaves similarly to a single-label SSVM in terms of efficient use of training data and computational effort during training, despite the increased complexity of the problem setting. In Section 4 we discuss MLSP's relation to existing methods for multi-label prediction with simple label sets, and to single-label structured prediction. We furthermore compare MLSP to a multi-label structured prediction methods within the SSVM framework in Section 4.1. In Section 5 we compare the different approaches experimentally, and we conclude in Section 6 by summarizing and discussing our contribution.

## 2 Multi-label structured prediction

We first recall some background and establish the notation necessary to discuss multi-label classification and structured prediction in a maximum margin framework. Our overall task is predicting outputs $y \in \mathcal{Y}$ for inputs $x \in \mathcal{X}$ in a supervised learning setting.

In ordinary *(single-label) multi-class prediction* we use a prediction function, $g : \mathcal{X} \to \mathcal{Y}$, for this, which we learn from i.i.d. example pairs $\{(x^i, y^i)\}_{i=1,\dots,n} \subset \mathcal{X} \times \mathcal{Y}$. Adopting a maximum-margin setting, we set

$$g(x) := \operatorname{argmax}_{y \in \mathcal{Y}} f(x, y) \quad \text{for a compatibility function} \quad f(x, y) := \langle w, \psi(x, y) \rangle. \quad (1)$$

The joint feature map $\psi : \mathcal{X} \times \mathcal{Y} \to \mathcal{H}$ maps input-output pairs into a Hilbert space $\mathcal{H}$ with inner product $\langle \cdot, \cdot \rangle$. It is defined either explicitly, or implicitly through a *joint kernel function* $k : (\mathcal{X} \times \mathcal{Y}) \times (\mathcal{X} \times \mathcal{Y}) \to \mathbb{R}$. We measure the quality of predictions by a task-dependent loss function $\Delta : \mathcal{Y} \times \mathcal{Y} \to \mathbb{R}^+$, where $\Delta(y, \bar{y})$ specifies what cost occurs if we predict an output $\bar{y}$ while the correct prediction is $y$.

*Structured output prediction* can be seen as a generalization of the above setting, where one wants to make not only one, but several dependent decisions at the same time, for example, deciding for each pixel of an image to which out of several semantic classes it belongs. Equivalently, one can interpret the same task as a special case of supervised single-label prediction, where inputs and outputs consist of multiple parts. In the above example, a whole image is one input sample, and a segmentation mask with as many entries as the image has pixels is an output. Having a choice of $M \geq 2$ classes per pixel of a $(w \times h)$-sized image leads to an output set of $M^{w \cdot h}$ elements. Enumerating all of these is out of question, and collecting training examples for each of them even more so. Consequently, structured output prediction requires specialized techniques that avoid enumerating all possible outputs, and that can generalize between labels in the output set. A popular technique for this task is the structured (output) support vector machine (SSVM) [3]. To train it, one has to solve a quadratic program subject to $n|\mathcal{Y}|$ linear constraints. If an efficient *separation oracle* is available, i.e. a technique for identifying the currently most violated linear constraints, *working set training*, in particular cutting plane [4] or bundle methods [5] allow SSVM training to arbitrary precision in polynomial time.

*Multi-label prediction* is a generalization of single-label prediction that gives up the condition of a functional relation between inputs and outputs. Instead, each input object can be associated with any (finite) number of outputs, including none. Formally, we are given pairs $\{(x^i, Y^i)\}_{i=1,\dots,n} \subset \mathcal{X} \times \mathbb{P}(\mathcal{Y})$, where $\mathbb{P}$ denotes the power set operation, and we want to determine a set-valued function $G : \mathcal{X} \to \mathbb{P}(\mathcal{Y})$. Often it is convenient to use *indicator vectors* instead of variable size subsets. We say that $v \in \{\pm 1\}^{\mathcal{Y}}$ represents the subset $Y \in \mathbb{P}(\mathcal{Y})$ if $v_y = +1$ for $y \in Y$ and $v_y = -1$ otherwise. Where no confusion arises, we use both representations interchangeably, e.g., we write either $Y^i$ or $v^i$ for a label set in the training data. To measure the quality of a predicted set we use a *set loss* function $\Delta_{ML} : \mathbb{P}(\mathcal{Y}) \times \mathbb{P}(\mathcal{Y}) \to \mathbb{R}$. Note that multi-label prediction can also be interpreted as ordinary single-output prediction with $\mathbb{P}(\mathcal{Y})$ taking the place of the original output set $\mathcal{Y}$. We will come back to this view in Section 4.1 when discussing related work.

*Multi-label structured prediction* combines the aspects of multi-label prediction and structured output sets: we are given a training set $\{(x^i, Y^i)\}_{i=1,\dots,n} \subset \mathcal{X} \times \mathbb{P}(\mathcal{Y})$, where $\mathcal{Y}$ is a structured output set of potentially very large size, and we would like to learn a prediction function: $G : \mathcal{X} \to \mathbb{P}(\mathcal{Y})$ with the ability to generalize also in the output set. In the following, we will take the view of structured prediction point of view, deriving expressions for predicting multiple structured outputs instead of single ones. Alternatively, the same conclusions could be reached by interpreting the task as performing multi-label predicting with binary output vectors that are too large to store or enumerate explicitly, but that have an internal structure allowing generalization between the elements.

## 3 Maximum margin multi-label structured prediction

In this section we propose a learning technique designed for multi-label structure prediction that we call MLSP. It makes set-valued prediction by[1],

$$G(x) := \{y \in \mathcal{Y} : f(x, y) > 0\} \quad \text{for} \quad f(x, y) := \langle w, \psi(x, y) \rangle. \quad (2)$$

Note that the compatibility function, $f(x, y)$, acts on individual inputs and outputs, as in single-label prediction (1), but the prediction step consists of collecting all outputs of positive scores instead of finding the outputs of maximal score. By including a constant entry into the joint feature map $\psi(x, y)$ we can model a bias term, thereby avoiding the need of a threshold during prediction (2). We can also add further flexibility by a data-independent, but label-dependent term. Note that our setup differs from SSVMs training in this regard. There, a bias term, or a constant entry of the feature map, would have no influence, because during training only pairwise differences of function values are considered, and during prediction a bias does not affect the argmax-decision in Equation (1).

We learn the weight vector $w$ for the MLSP compatibility function in a maximum-margin framework that is derived from regularized risk minimization. As the risk depends on the loss function chosen, we first study the possibilities we have for the set loss $\Delta_{ML} : \mathbb{P}(\mathcal{Y}) \times \mathbb{P}(\mathcal{Y}) \to \mathbb{R}^+$. There are no established functions for this in the structured prediction setting, but it turns out that two canonical set losses are consistent with the following first principles. *Positivity:* $\Delta_{ML}(Y, \bar{Y}) \geq 0$, with equality only if $Y = \bar{Y}$, *Modularity:* $\Delta_{ML}$ should decompose over the elements of $\mathcal{Y}$ (in order to facilitate efficient computation), *Monotonicity:* $\Delta_{ML}$ should reflect that making a wrong decision about some element $y \in \mathcal{Y}$ can never reduce the loss. The last criterion we formalize as

$$\Delta_{ML}(Y, \bar{Y} \cup \{\bar{y}\}) \geq \Delta_{ML}(Y, \bar{Y}) \quad \text{for any } \bar{y} \notin Y, \text{ and} \tag{3}$$

$$\Delta_{ML}(Y \cup \{y\}, \bar{Y}) \geq \Delta_{ML}(Y, \bar{Y}) \quad \text{for any } y \notin \bar{Y}. \tag{4}$$

Two candidates that fulfill these criteria are the *sum loss*, $\Delta_{sum}(Y, \bar{Y}) := \sum_{y \in Y \ominus \bar{Y}} \lambda(Y, y)$, and the *max loss*, $\Delta_{max}(Y, \bar{Y}) := \max_{y \in Y \ominus \bar{Y}} \lambda(Y, y)$, where $Y \ominus \bar{Y} := (Y \backslash \bar{Y}) \cup (\bar{Y} \backslash Y)$ is the symmetric set difference, and $\lambda : \mathbb{P}(\mathcal{Y}) \times \mathcal{Y} \to \mathbb{R}^+$ is a task-dependent per-label misclassification cost. Assuming that a set $Y$ is the correct prediction, $\lambda(Y, \bar{y})$ specifies either the cost of predicting $\bar{y}$, although $\bar{y} \notin Y$, or of not predicting $\bar{y}$, when really $\bar{y} \in Y$. In the special case of $\lambda \equiv 1$ the sum loss is known as *symmetric difference loss*, and it coincides with the *Hamming loss* of the binary indicator vector representation. The max loss becomes the $0/1$-loss between sets in this case. In a general case, $\lambda$ typically expresses partial correctness, generalizing the single-label structured loss $\Delta(y, \bar{y})$. Note that in evaluating $\lambda(Y, \bar{y})$ one has access to the whole set $Y$, not just single elements. Therefore, a flexible penalization of multiple errors is possible, e.g., *submodular* behavior.

While in the small-scale multi-label situation, the sum loss is more common, we argue in this work that that the max loss has advantages in the structured prediction situation. For once, the sum loss has a scaling problem. Because it adds potentially exponentially many terms, the ratio in loss between making few mistakes and making many mistakes is very large. If used in the unnormalized form given above this can result in impractically large values. Normalizing the expression by multiplying with $1/|\mathcal{Y}|$ stabilizes the upper value range, but it leads to a situation where $\Delta_{sum}(Y, \bar{Y}) \approx 0$ in the common situation that $\bar{Y}$ differs from $Y$ in only a few elements. The value range of the max loss, on the other hand, is the same as the value range of $\lambda$ and therefore easy to keep reasonable. A second advantage of the max loss is that it leads to an efficient constraint generation technique during training, as we will see in Section 3.3.

## 3.1 Maximum margin multi-label structured prediction (MLSP)

To learn the parameters $w$ of the compatibility function $f(x, y)$ we follow a regularized risk minimization framework: given i.i.d. training examples $\{(x^i, Y^i)\}_{i=1,\dots,n}$, we would like to minimize $\frac{1}{2}\|w\|^2 + \frac{C}{n}\sum_i \Delta_{max}(Y^i, G(x^i))$. Using the definition of $\Delta_{max}$ this is equivalent to minimizing $\frac{1}{2}\|w\|^2 + \frac{C}{n}\sum_i \xi^i$, subject to $\xi^i \geq \lambda(Y^i, y)$ for all $y \in \mathcal{Y}$ with $v_y^i f(x^i, y) \leq 0$. Upper bounding the inequalities by a Hinge construction yields the following maximum-margin training problem:

$$(w^*, \xi^*) = \operatorname*{argmin}_{w \in \mathcal{H}, \xi^1, \dots, \xi^n \in \mathbb{R}^+} \frac{1}{2}\|w\|^2 + \frac{C}{n}\sum_{i=1}^{n} \xi^i \tag{5}$$

subject to, for $i = 1, \dots, n$,

$$\xi^i \geq \lambda(Y^i, y)[1 - v_y^i f(x^i, y)], \text{ for all } y \in \mathcal{Y}. \tag{6}$$

---

Note that making per-label decisions through thresholding does not rule out the sharing of information between labels. In the terminology of [7], Equation (2) corresponds to a *conditional label independence* assumption. Through the joint feature function $\psi(x, y)$ te proposed model can still learn *unconditional dependence* between labels, which relates closer to an intuition of the form "Label $A$ tends to co-occur with label $B$".

Besides this *slack rescaled* variant, one can also form *margin rescaled* training using the constraints

$$\xi^i \geq \lambda(Y^i, y) - v^i_y f(x^i, y), \quad \text{for all } y \in \mathcal{Y}. \tag{7}$$

Both variants coincide in the case of $0/1$ set loss, $\lambda(Y^i, y) \equiv 1$. The main difference between slack and margin rescaled training is how they treat the case of $\lambda(Y^i, y) = 0$ for some $y \in \mathcal{Y}$. In slack rescaling, the corresponding outputs have no effect on the training at all, whereas for margin rescaling, no margin is enforced for such examples, but a penalization still occurs whenever $f(x^i, y) > 0$ for $y \notin Y^i$, or if $f(x^i, y) < 0$ for $y \in Y^i$.

## 3.2 Generalization Properties

Maximum margin structured learning has become successful not only because it provides a powerful framework for solving practical prediction problems, but also because it comes with certain theoretical guarantees, in particular generalization bounds. We expect that many of these results will have multi-label analogues. As an initial step, we formulate and prove a generalization bound for slack-rescaled MLSP similar to the single-label SSVM analysis in [8].

Let $G_w(x) := \{y \in \mathcal{Y} : f_w(x, y) > 0\}$ for $f_w(x, y) = \langle w, \psi(x, y) \rangle$. We assume $|\mathcal{Y}| < r$ and $\|\psi(x, y)\| < s$ for all $(x, y) \in \mathcal{X} \times \mathcal{Y}$, and $\lambda(Y, y) \leq \Lambda$ for all $(Y, y) \in \mathbb{P}(\mathcal{Y}) \times \mathcal{Y}$.

For any distribution $Q_w$ over weight vectors, that may depend on $w$, we denote by $L(Q_w, P)$ the expected $\Delta_{max}$-risk for $P$-distributed data,

$$L(Q_w, P) = \mathbb{E}_{\bar{w} \sim Q_w} \{ \mathcal{R}_{P, \Delta_{max}}(G_{\bar{w}}) \} = \mathbb{E}_{\bar{w} \sim Q_w, (x, Y) \sim P} \{ \Delta_{max}(Y, G_{\bar{w}}(x)) \}. \tag{8}$$

The following theorem bounds the expected risk in terms of the total margin violations.

**Theorem 1.** *With probability at least $1 - \sigma$ over the sample $S$ of size $n$, the following inequality holds simultaneously for all weight vectors $w$.*

$$L(Q_w, D) \leq \frac{1}{n} \sum_{i=1}^{n} \ell(x^i, Y^i, f) + \frac{\|w\|^2}{n} + \left( \frac{s^2 \|w\|^2 \ln(rn/\|w\|^2) + \ln \frac{n}{\sigma}}{2(n-1)} \right)^{1/2} \tag{9}$$

*for $\ell(x^i, Y^i, f) := \max_{y \in \mathcal{Y}} \lambda(Y^i, y) [\![ v^i_y f(x^i, y) < 1 ]\!]$, where $v^i$ is the binary indicator vector of $Y^i$.*

*Proof.* The argument follows [8, Section 11.6]. It can be found in the supplemental material. □

A main insight from Theorem 1 is that the number of samples needed for good generalization grows only logarithmically with $r$, i.e. the size of $\mathcal{Y}$. This is the same complexity as for single-label prediction using SSVMs, despite the fact that multi-label prediction formally maps into $\mathbb{P}(\mathcal{Y})$, i.e. an exponentially larger output set.

## 3.3 Numeric Optimization

The numeric solution of MLSP training resembles SSVM training. For explicitly given joint feature maps, $\psi(x, y)$, we can solve the optimization problem (5) in the primal, for example using subgradient descent. To solve MLSP in a kernelized setup we introduce Lagrangian multipliers $(\alpha^i_y)_{i=1,\dots,n; y \in \mathcal{Y}}$ for the constraints (7)/(6). For the margin-rescaled variant we obtain the dual

$$\max_{\alpha^i_y \in \mathbb{R}^+} -\frac{1}{2} \sum_{(i,y),(\bar{\imath},\bar{y})} v^i_y v^{\bar{\imath}}_{\bar{y}} \alpha^i_y \alpha^{\bar{\imath}}_{\bar{y}} \, k\big((x^i, y), (x^{\bar{\imath}}, \bar{y})\big) + \sum_{(i,y)} \lambda^i_y \alpha^i_y \tag{10}$$

$$\text{subject to} \quad \sum_y \alpha^i_y \leq \frac{C}{n}, \quad \text{for } i = 1, \dots, n. \tag{11}$$

For slack-rescaled MLSP, the dual is computed analogously as

$$\max_{\alpha^i_y \in \mathbb{R}^+} -\frac{1}{2} \sum_{(i,y),(\bar{\imath},\bar{y})} v^i_y v^{\bar{\imath}}_{\bar{y}} \alpha^i_y \alpha^{\bar{\imath}}_{\bar{y}} \, k\big((x^i, y), (x^{\bar{\imath}}, \bar{y})\big) + \sum_{(i,y)} \alpha^i_y \tag{12}$$

$$\text{subject to} \quad \sum_y \frac{\alpha^i_y}{\lambda^i_y} \leq \frac{C}{n}, \quad \text{for } i = 1, \dots, n, \tag{13}$$

with the convention that only terms with $\lambda_y^i \neq 0$ enter the summation. In both cases, the compatibility function becomes

$$f(x, y) = \sum_{(i, \bar{y})} \alpha_{\bar{y}}^i v_{\bar{y}}^i \, k\big((x^i, \bar{y}), (x, y)\big). \tag{14}$$

Comparing the optimization problems (10)/(11) and (12)/(13) to the ordinary SVM dual, we see that MLSP couples $|\mathcal{Y}|$ binary SVM problems by the joint kernel function and the summed-over box constraints. In particular, whenever only a feasibly small subset of variables has to be considered, we can solve the problem using a general purpose QP solver, or a slightly modified SVM solver. Overall, however, there are infeasibly many constraints in the primal, or variables in the dual. Analogously to the SSVM situation we therefore apply iterative working set training, which we explain here using the terminology of the primal. We start with an arbitrary, e.g. empty, working set $S$. Then, in each step we solve the optimization using only the constraints indicated by the working set. For the resulting solution $(w_S, \xi_S)$ we check whether any constraints of the full set (6)/(7) are violated up to a target precision $\epsilon$. If not, we have found the optimal parameters. Otherwise, we add the most violated constraint to $S$ and start the next iteration. The same monotonicity argument as in [3] shows that we reach an objective value $\epsilon$-close to the optimal one within $O(\frac{1}{\epsilon})$ steps. Consequently, MLSP training is roughly comparable in computational complexity to SSVM training.

The crucial step in working set training is the identification of violated constraints. Note that constraints in MLSP are determined by pairs of samples and single labels, not pairs of samples and sets of labels. This allows us to reuse existing methods for loss augmented single label inference. In practice, it is safe to assume that the sets $Y^i$ are feasibly small, since they are given to us explicitly. Consequently, we can identify violated "positive" constraints by explicitly checking the inequalities (7)/(6) for $y \in Y^i$. Identifying violated "negative" constraint requires loss-augmented prediction over $\mathcal{Y} \setminus Y^i$. We are not aware of a general purpose solution for this task, but at least all problems that allow efficient $K$-best MAP prediction can be handled by iteratively performing loss-augmented prediction within $\mathcal{Y}$ until a violating example from $\mathcal{Y} \setminus Y^i$ is found, or it is confirmed that no such example exists. Note that $K$-best versions of most standard MAP prediction methods have been developed, including max-flow [9], loopy BP [10], LP-relaxations [11], and Sampling [12].

### 3.4 Prediction problem

After training, Equation (2) specifies the rule to predict output sets for new input data. In contrast to single-label SSVM prediction this requires not only a maximization over all elements of $\mathcal{Y}$, but the collection of all elements $y \in \mathcal{Y}$ of positive score. The structure of the output set is not as immediately helpful for this as it is, e.g., in MAP prediction. Task-specific solutions exist, however, for example branch-and-bound search for object detection [13]. Also, it is often possible to establish an upper bound on the number of desired outputs, and then, $K$-best prediction techniques can again be applied. This makes MLSP of potential use for several classical tasks, such as *parsing* and *chunking* in natural language processing, *secondary structured prediction* in computational biology, or *human pose estimation* in computer vision. In general situations, evaluating (2) might require approximate structured prediction techniques, e.g. iterative greedy selection [14]. Note that the use of approximation algorithms is little problematic here, because, in contrast to training, the prediction step is not performed in an iterative manner, so errors do not accumulate.

## 4 Related Work

Multi-label classification is an established field of research in machine learning and several established techniques are available, most of which fall into one of three categories: 1) *Multi-class reformulations* [15] treat every possible label subset, $Y \in \mathbb{P}(\mathcal{Y})$, as a new class in an independent multi-class classification scenario. 2) *Per-label decomposition* [16] trains one classifier for each output label and makes independent decision for each of those. 3) *Label ranking* [17] learns a function that ranks all potential labels for an input sample. Given the size of $\mathcal{Y}$, 1) is not a promising direction for multi-label structured prediction. A straight-forward application of 2) and 3) are also infeasible if $\mathcal{Y}$ is too large to enumerate. However, MLSP resembles both approaches by sharing their prediction rule (2). MLSP can be seen as a way to make a combination of approaches applicable to the situation of structured prediction by incorporating the ability to generalize in the label set.

Besides the general concepts above, many specific techniques for multi-label prediction have been proposed, several of them making use of structured prediction techniques: [18] introduces an SSVM

formulation that allows direct optimization of the average precision ranking loss when the label set can be enumerated. [19] relies on a counting framework for this purpose, and [20] proposes an SSVM formulation for enforcing diversity between the labels. [21] and [22] identify shared subspaces between sets of labels, [23] encodes linear label relations by a change of the SSVM regularizer, and [24] handles the case of tree- and DAG-structured dependencies between possible outputs. All these methods work in the multi-class setup and require an explicit enumerations of the label set. They use a structured prediction framework to encode dependencies between the individual output labels, of which there are relatively few. MLSP, on the other hand, aims at predicting multiple structured object, i.e. the structured prediction framework is not just a tool to improve multi-class classification with multiple output labels, but it is required as a core component for predicting even a single output.

Some previous methods targeting multi-label prediction with large output sets, in particular using *label compression* [25] or a *label hierarchy* [26]. This allows handling thousands of potential output classes, but a direct application to the structured prediction situation is not possible, because the methods still require explicit handling of the output label vectors, or cannot predict labels that were not part of the training set.

The actual task of predicting multiple structured outputs has so far not appeared explicitly in the literature. The situation of multiple inputs during training has, however, received some attention: [27] introduces a one-class SVM based training technique for learning with ambiguous ground truth data. [13] trains an SSVM for the same task by defining a task-adapted loss function $\Delta^{min}(Y, \bar{y}) = \min_{y \in Y} \Delta(y, \bar{y})$. [28] uses a similar min-loss in a CRF setup to overcome problems with incomplete annotation. Note that $\Delta^{min}(Y, \bar{y})$ has the right signature to be used as a misclassification cost $\lambda(Y, \bar{y})$ in MLSP. The compatibility functions learned by the maximum-margin techniques [13, 27] have the same functional form as $f(x, y)$ in MLSP, so they can, in principle, be used to predict multiple outputs using Equation (2). However, our experiments of Section 5 show that this leads to low multi-label prediction accuracy, because the training setup is not designed for this evaluation procedure.

### 4.1 Structured Multilabel Prediction in the SSVM Framework

At first sight, it appears unnecessary to go beyond the standard structured prediction framework at all in trying to predict subsets of $\mathcal{Y}$. As mentioned in Section 3, multi-label prediction into $\mathcal{Y}$ can be interpreted as single-label prediction into $\mathbb{P}(\mathcal{Y})$, so a straight-forward approach to multi-label structured prediction would be to use an ordinary SSVM with output set $\mathbb{P}(\mathcal{Y})$. We will call this setup $\mathbb{P}$-SSVM. It has previously been proposed for classical multi-label prediction, for example in [23]. Unfortunately, as we will show in this section, the $\mathbb{P}$-SSVM setup is not well suited to the structured prediction situation.

A $\mathbb{P}$-SSVM learns a prediction function, $G(x) := \mathrm{argmax}_{Y \in \mathbb{P}(\mathcal{Y})} F(x, Y)$, with linearly parameterized compatibility function, $F(x, Y) := \langle w, \psi(x, Y) \rangle$, by solving the optimization problem

$$\underset{w \in \mathcal{H}, \xi^1, \ldots, \xi^n \in \mathbb{R}^+}{\mathrm{argmin}} \frac{1}{2} \|w\|^2 + \frac{C}{n} \sum_{i=1}^{n} \xi^i, \quad \text{subject to} \quad \xi^i \geq \Delta(y^i, Y) + F(x^i, Y) - F(x^i, Y^i), \quad (15)$$

for $i = 1, \ldots, n$, and for all $Y \in \mathbb{P}(\mathcal{Y})$. The main problem with this general form is that identifying violated constraints of (15) requires loss-augmented maximization of $F$ over $\mathbb{P}(\mathcal{Y})$, i.e. an exponentially larger set than $\mathcal{Y}$. To better understand this problem, we analyze what happens when making the same simplifying assumptions as for MLSP in Section 3.1. First, we assume additivity of $F$ over $\mathcal{Y}$, i.e. $F(x, Y) := \sum_{y \in Y} f(x, y)$ for $f(x, y) := \langle w, \psi(x, y) \rangle$. This turns the argmax-evaluation for $G(x)$ exactly into the prediction rule (2), and the constraint set in (15) simplifies to

$$\xi^i \geq \Delta_{ML}(Y^i, Y) - \sum_{y \in Y \ominus Y^i} v_y^i f(x^i, y), \qquad \text{for } i = 1, \ldots, n, \text{ and for all } Y \in \mathbb{P}(\mathcal{Y}), \quad (16)$$

Choosing $\Delta_{ML}$ as max loss does not allow us to further simplify this expression, but choosing the sum loss does: with $\Delta_{ML}(Y^i, Y) = \sum_{y \in Y \ominus Y^i} \lambda(Y^i, y)$, we obtain an explicit expression for the label set maximizing the right hand side of the constraint (16), namely

$$Y_{\mathrm{viol}}^i = \{y \in Y^i : f(x^i, y) < \lambda(Y^i, y)\} \cup \{y \in \mathcal{Y} \setminus Y^i : f(x^i, y) > -\lambda(Y^i, y)\}. \quad (17)$$

Thus, we avoid having to maximize a function over $\mathbb{P}(\mathcal{Y})$. Unfortunately, the set $Y_{\mathrm{viol}}^i$ in Equation (17) can contain exponentially many terms, rendering a numeric computation of $F(x^i, Y_{\mathrm{viol}}^i)$ or

its gradient still infeasible in general. Note that this is not just a rare, easily avoidable case. Because $w$, and thereby $f$, are learned iteratively, they typically go through phases of low prediction quality, i.e. large $Y_{\text{viol}}^i$. In fact, starting the optimization with $w = 0$ would already lead to $Y_{\text{viol}}^i = \mathcal{Y}$ for all $i = 1, \ldots, n$. Consequently, we presume that $\mathbb{P}$-SSVM training is intractable for structured prediction problems, except for the case of a small label set.

Note that while computational complexity is the most prominent problem of $\mathbb{P}$-SSVM training, it is not the only one. For example, even if we did find a polynomial-time training algorithm to solve (15) the generalization ability of the resulting predictor would be unclear: the SSVM-generalization bounds [8] suggest that training sets of size $O(\log |\mathbb{P}(\mathcal{Y})|) = O(|\mathcal{Y}|)$ will be required, compared to the $O(\log |\mathcal{Y}|)$ bound we established for MLSP in Section 3.2.

## 5 Experimental Evaluation

To show the practical use of MLSP we performed experiments on *multi-label hierarchical classification* and *object detection in natural images*. The complete protocol of training a miniature toy example can be found in the supplemental material (available from the author's homepage).

### 5.1 Multi-label hierarchical classification

We use *hierarchical classification* as an illustrative example that in particular allows us to compare MLSP to alternative, less scalable, methods. On the one hand, it is straight-forward to model as a structured prediction task, see e.g. [3, 29, 30, 31]. On the other hand, its output set is small enough such that we can compare MLSP also against other approaches that cannot handle very large output sets, in particular $\mathbb{P}$-SSVM and independent per-label training.

The task in hierarchical classification is to classify samples into a number of discrete classes, where each class corresponds to a path in a tree. Classes are considered related if they share a path in the tree, and this is reflected by sharing parts of the joint feature representations. In our experiments, we use the *PASCAL VOC2006* dataset that contains 5304 images, each belonging to between 1 and 4 out of 10 classes. We represent each image $x$ by 960-dimensional GIST features $\phi(x)$ and use the same 19-node hierarchy $\kappa$ and joint feature function, $\psi(x, y) = vec(\phi(x) \otimes \kappa(y))$, as in [30]. As baselines we use $\mathbb{P}$-SSVM [23], JKSE [27], and an SSVM trained with the normal, single-label objective, but evaluated by Equation (2). We follow the pre-defined data splits, doing model selection using the *train* and *val* parts to determine $C \in \{2^{-1}, \ldots, 2^{14}\}$ (MLSP, $\mathbb{P}$-SSVM, SSVM), or $\nu \in \{0.05, 0.10, \ldots, 0.95\}$ (JKSE). We then retrain on the combination of *train* and *val* and we test on the *test* part of the dataset. As the label set is small, we use exhaustive search over $\mathcal{Y}$ to identify violated constraints during training and to perform the final predictions.

We report results in Table 1a). As there is no single established multi-label error measure, and because it illustrates the effect of training with different loss function, we report several common measures. The results show nicely how the assumptions made during training influence the prediction characteristics. Qualitatively, MLSP achieves best prediction accuracy in the max loss, $\mathbb{P}$-SSVM is better if we judge by the sum loss. This exactly reflects the loss functions they are trained with. Independent training achieves very good results with respect to both measures, justifying its common use for multi-label prediction with small label sets and many training examples per label[2] Ordinary SSVM training does not achieve good max- or sum-loss scores, but it performs well if quality is measured by the average of the area under the precision-recall curves across labels for each individual test example. This is also plausible, as SSVM training uses a ranking-like loss: all potential labels for each input are enforced to be in the right order (correct labels have higher score than incorrect ones), but nothing in the objective encourages a cut-off point at $0$. As a consequence, too few or too many labels are predicted by Equation (2). In Table 1a) it appears to be too many, visible as high recall but low precision. JKSE does not achieve competitive results in max loss, mAUC loss or F1-score. Potentially this is because we use it with a linear kernel to stay comparable with the other methods, whereas [27] reported good results mainly for nonlinear kernels.

Qualitatively, MLSP and $\mathbb{P}$-SSVM show comparable prediction quality. We take this as an indication that both, training with sum loss and training with max loss, make sense conceptually. However, of

Figure 1: Multi-label structured prediction results. $\Delta_{max}/\Delta_{sum}$: max/sum loss (lower is better), mAUC: mean area under per-sample precision-recall curve, prec/rec/F1: precision, recall, F1-score (higher is better). Methods printed in *italics* are infeasible for general structured output sets.

| | $\Delta_{max}$ | $\Delta_{sum}$ | mAUC | F1 ( prec / rec ) |
|---|---|---|---|---|
| MLSP | 0.73 | 1.59 | 0.82 | 0.42 ( 0.40 / 0.46 ) |
| JKSE | 1.00 | 1.91 | 0.54 | 0.23 ( 0.14 / 0.76 ) |
| SSVM | 0.88 | 3.86 | 0.84 | 0.37 ( 0.24 / 0.88 ) |
| *$\mathbb{P}$-SSVM* | *0.75* | *1.11* | *0.83* | *0.44 ( 0.48 / 0.41 )* |
| *indep.* | *0.73* | *1.07* | *0.84* | *0.46 ( 0.61 / 0.38 )* |

(a) Hierarchical classification results.

| | $\Delta_{max}$ | $\Delta_{sum}$ | F1 ( prec / rec ) |
|---|---|---|---|
| MLSP | 0.66 | 1.31 | 0.46 ( 0.60 / 0.52 ) |
| JKSE | 0.99 | 7.29 | 0.09 ( 0.60 / 0.16 ) |
| SSVM | 0.93 | 3.71 | 0.21 ( 0.79 / 0.33 ) |
| *$\mathbb{P}$-SSVM* | *infeasible* | | |
| *indep.* | *infeasible* | | |

(b) Object detection results.

the five methods, only MLSP, JKSE and SSVM generalize to more general structured prediction setting, as they do not require exhaustive enumeration of the label set. Amongst these, MLSP is preferable, except if one is only interested in ranking the labels, for which SSVM also works well.

## 5.2 Object class detection in natural images

Object detection can be solved as a structured prediction problem where natural images are the inputs and coordinate tuples of bounding boxes are the outputs. The label set is of quadratic size in the number of image pixels and thus cannot be searched exhaustively. However, efficient (loss-augmented) $\mathrm{argmax}$-prediction can be performed by branch-and-bound search [33]. Object detection is also inherently a multi-label task, because natural images contain different numbers of objects. We perform experiments on the public UIUC-Cars dataset [34]. Following the experimental setup of [27] we use the *multiscale* part of the dataset for training and the *singlescale* part for testing. The additional set of pre-cropped car and background images serves as validation set for model selection. We use the *localization kernel*, $k\big((x,y),(\bar{x},\bar{y})\big) = \phi(x|_y)^t \phi(\bar{x}|_{\bar{y}})$ where $\phi(x|_y)$ is a 1000-dimensional bag of visual words representation of the region $y$ within the image $x$ [13]. As misclassification cost we use $\lambda(Y,y) := 1$ for $y \in Y$, and $\lambda(Y,y) := \min_{\bar{y} \in Y} A(\bar{y}, y)$ otherwise, where $A(\bar{y}, y) := 0$ if $\frac{\text{area}(\bar{y} \cap y)}{\text{area}(\bar{y} \cup y)} \geq 0.5$, and $A(\bar{y}, y) := 1$ otherwise. This is a common measure in object detection, which reflects the intuition that all objects in an image should be identified, and that an object's position is acceptable if it overlaps sufficiently with at least one ground truth object. $\mathbb{P}$-SSVM and independent training are not applicable in this setup, so we compare MLSP against JKSE and SSVM. For each method we train models on the training set and choose the $C$ or $\nu$ value that maximizes the F1 score over the validation set of precropped object and background images. Prediction is performed using branch and bound optimization with greedy non-maximum suppression [35]. Table 1b) summarizes the results on the test set (we do not report the *mAUC* measure, as computing this would require summing over the complete output set). One sees that MLSP achieves the best results amongst the three method. SSVM as well as JKSE suffer particularly from low recall, and their predictions also have higher sum loss as well as max loss.

## 6 Summary and Discussion

We have studied multi-label classification for structured output sets. Existing multi-label techniques cannot directly be applied to this task because of the large size of the output set, and our analysis showed that formulating multi-label structured prediction set a set-valued structured support vector machine framework also leads to infeasible training problems. Instead, we proposed an new maximum-margin formulation, MLSP, that remains computationally tractable by use of the max loss instead of sum loss between sets, and shows several of the advantageous properties known from other maximum-margin based techniques, in particular a convex training problem and PAC-Bayesian generalization bounds. Our experiments showed that MLSP has higher prediction accuracy than baseline methods that remain applied in structured output settings. For small label sets, where both concepts are applicable, MLSP performs comparable to the set-valued SSVM formulation.

Besides these promising initial results, we believe that there are still several aspects of multi-label structured prediction that need to be better understood, in particular the prediction problem at test time. Collecting all elements of positive score is a natural criterion, but it is costly to perform exactly if the output set is very large. Therefore, it would be desirable to develop sparsity enforcing variations of Equation (2), for example by adopting ideas from compressed sensing [25].

## Footnotes

[1] More complex prediction rules exist in the multi-label literature, see, e.g., [6]. We restrict ourselves to per-label thresholding, because more advanced rules complicate the learning and prediction problem even further.

[2]For $\Delta_{sum}$ this is not surprising: independent training is known to be the optimal setup, if enough data is available [32]. For $\Delta_{sum}$, the *multi-class reformulation* would be the optimal setup. The problem in multi-label structured prediction is solely that $|\mathcal{Y}|$ is too large, and training data too scarce, to use either of these setups.

# References

[1] J. D. Lafferty, A. McCallum, and F. C. N. Pereira. Conditional random fields: Probabilistic models for segmenting and labeling sequence data. In *ICML*, 2001.

[2] B. Taskar, C. Guestrin, and D. Koller. Max-margin Markov networks. In *NIPS*, 2003.

[3] I. Tsochantaridis, T. Joachims, T. Hofmann, and Y. Altun. Large margin methods for structured and interdependent output variables. *JMLR*, 6, 2006.

[4] T. Joachims, T. Finley, and C. N. J. Yu. Cutting-plane training of structural SVMs. *Machine Learning*, 77(1), 2009.

[5] C. H. Teo, SVN Vishwanathan, A. Smola, and Q. V. Le. Bundle methods for regularized risk minimization. *JMLR*, 11, 2010.

[6] G. Tsoumakas and I. Katakis. Multi-label classification: An overview. *International Journal of Data Warehousing and Mining*, 3(3), 2007.

[7] K. Dembczynski, W. Cheng, and E. Hüllermeier. Bayes optimal multilabel classification via probabilistic classifier chains. In *ICML*, 2011.

[8] D. McAllester. Generalization bounds and consistency for structured labeling. In G. Bakır, T. Hofmann, B. Schölkopf, A.J. Smola, and B. Taskar, editors, *Predicting Structured Data*. MIT Press, 2007.

[9] D. Nilsson. An efficient algorithm for finding the M most probable configurations in probabilistic expert systems. *Statistics and Computing*, 8(2), 1998.

[10] C. Yanover and Y. Weiss. Finding the M most probable configurations using loopy belief propagation. In *NIPS*, 2004.

[11] M. Fromer and A. Globerson. An LP View of the M-best MAP problem. In *NIPS*, 2009.

[12] J. Porway and S.-C. Zhu. $C^4$: Exploring multiple solutions in graphical models by cluster sampling. *PAMI*, 33(9), 2011.

[13] M. B. Blaschko and C. H. Lampert. Learning to localize objects with structured output regression. In *ECCV*, 2008.

[14] A. Bordes, N. Usunier, and L. Bottou. Sequence labelling SVMs trained in one pass. *ECML PKDD*, 2008.

[15] M. R. Boutell, J. Luo, X. Shen, and C.M. Brown. Learning multi-label scene classification. *Pattern Recognition*, 37(9), 2004.

[16] T. Joachims. Text categorization with support vector machines: Learning with many relevant features. In *ECML*, 1998.

[17] R. E. Schapire and Y. Singer. Boostexter: A boosting-based system for text categorization. *Machine Learning*, 39(2–3), 2000.

[18] Y. Yue, T. Finley, F. Radlinski, and T. Joachims. A support vector method for optimizing average precision. In *ACM SIGIR*, 2007.

[19] T. Gärtner and S. Vembu. On structured output training: Hard cases and an efficient alternative. *Machine Learning*, 76(2):227–242, 2009.

[20] Y. Yue and T. Joachims. Predicting diverse subsets using structural SVMs. In *ICML*, 2008.

[21] S. Ji, L. Tang, S. Yu, and J. Ye. Extracting shared subspaces for multi-label classification. In *ACM SIGKDD*, 2008.

[22] P. Rai and H. Daumé III. Multi-label prediction via sparse infinite CCA. In *NIPS*, 2009.

[23] B. Hariharan, L. Zelnik-Manor, S. V. N. Vishwanathan, and M. Varma. Large scale max-margin multi-label classification with priors. In *ICML*, 2010.

[24] W. Bi and J. Kwok. Multi-label classification on tree- and DAG-structured hierarchies. In *ICML*, 2011.

[25] D. Hsu, S. Kakade, J. Langford, and T. Zhang. Multi-label prediction via compressed sensing. In *NIPS*, 2009.

[26] G. Tsoumakas, I. Katakis, and I. Vlahavas. Effective and efficient multilabel classification in domains with large number of labels. In *ECML PKDD*, 2008.

[27] C. H. Lampert and M. B. Blaschko. Structured prediction by joint kernel support estimation. *Machine Learning*, 77(2–3), 2009.

[28] J. Petterson, T. S. Caetano, J. J. McAuley, and J. Yu. Exponential family graph matching and ranking. In *NIPS*, 2009.

[29] J. Rousu, C. Saunders, S. Szedmak, and J. Shawe-Taylor. Kernel-based learning of hierarchical multilabel classification models. *JMLR*, 7, 2006.

[30] A. Binder, K.-R. Müller, and M. Kawanabe. On taxonomies for multi-class image categorization. *IJCV*, 2011.

[31] L. Cai and T. Hofmann. Hierarchical document categorization with support vector machines. In *ICKM*, 2004.

[32] K. Dembczynski, W. Cheng, and E. Hüllermeier. Bayes optimal multilabel classification via probabilistic classifier chains. In *ICML*, 2010.

[33] C. H. Lampert, M. B. Blaschko, and T. Hofmann. Efficient subwindow search: A branch and bound framework for object localization. *PAMI*, 31(12), 2009.

[34] S. Agarwal, A. Awan, and D. Roth. Learning to detect objects in images via a sparse, part-based representation. *PAMI*, 26(11), 2004.

[35] C. H. Lampert. An efficient divide-and-conquer cascade for nonlinear object detection. In *CVPR*, 2010.

